# Proper losses for learning from partial labels

**Jesús Cid-Sueiro**
Department of Signal Theory and Communications
Universidad Carlos III de Madrid
Legans-Madrid, 28911 Spain
`jcid@tsc.uc3m.es`

## Abstract

This paper discusses the problem of calibrating posterior class probabilities from partially labelled data. Each instance is assumed to be labelled as belonging to one of several candidate categories, at most one of them being true. We generalize the concept of proper loss to this scenario, we establish a necessary and sufficient condition for a loss function to be proper, and we show a direct procedure to construct a proper loss for partial labels from a conventional proper loss. The problem can be characterized by the mixing probability matrix relating the true class of the data and the observed labels. The full knowledge of this matrix is not required, and losses can be constructed that are proper for a wide set of mixing probability matrices.

## 1 Introduction

The problem of learning multiple classes from data with imprecise label information has attracted a recent attention in the literature. It arises in many different applications: Cour [1] cites some of them: picture collections containing several faces per image and a caption that only specifies who is in the picture but not which name matches which face, or video collections with labels taken from annotations.

In a partially labelled data set, each instance is assigned to a set of candidate categories, at most only one of them true. The problem is closely related to learning from noisy labels, which is common in human-labelled data bases with multiple annotators [2] [3] medical imaging, crowdsourcing, etc. Other related problems can be interpreted as particular forms of partial labelling: semisupervised learning, or hierarchical classification in databases where some instances could be labelled with respect to parent categories only. It is also a particular case of the more general problems of learning from soft labels [4] or learning from measurements [5].

Several algorithms have been proposed to deal with partial labelling [1] [2] [6] [7] [8]. Though some theoretical work has been addressed in order to analyze the consistency of algorithms [1] or the information provided by uncertain data [8], little effort has been done to analyze the conditions under which the true class can be inferred from partial labels.

In this paper we address the problem of estimating posterior class probabilities from partially labelled data. In particular, we obtain general conditions under which the posterior probability of the true class given the observation can be estimated from training data with ambiguous class labels. To do so, we generalize the concept of proper losses to losses that are functions of ambiguous labels, and show that the capability to estimate posterior class probabilities using a given loss depends on the probability matrix relating the ambiguous labels with the true class of the data. Each generalized proper loss can be characterized by the set (a convex polytope) of all admissible probability matrices. Analyzing the structure of these losses is one of the main goals of this paper. Up to our knowledge, the design of proper losses for learning from imperfect labels has not been addressed in the area of Statistical Learning.

The paper is organized as follows: Sec. 2 formulates the problem discussed in the paper, Sec. 3 generalizes proper losses to scenarios with ambiguous labels, Sec. 4 proposes a procedure to design proper losses for wide sets of mixing matrices, Sec. 5 discusses estimation errors and Sec. 6 states some conclusions.

## 2 Formulation

### 2.1 Notation

Vectors are written in boldface, matrices in boldface capital and sets in calligraphic letters. For any integer $n$, $\mathbf{e}_i^n$ is a $n$-dimensional unit vector with all zero components apart from the $i$-th component which is equal to one, and $\mathbb{1}_n$ is a $n$-dimensional all-ones vector. Superindex $^\mathrm{T}$ denotes transposition. We will use $\ell()$ to denote a loss based on partial labels, and $\tilde{\ell}$ to losses based on true labels. The simplex of $n$-dimensional probability vectors is $\mathcal{P}_n = \{\mathbf{p} \in [0,1]^n : \sum_{i=0}^{n-1} p_i = 1\}$ and the set of all left-stochastic matrices is $\mathcal{M} = \{\mathbf{M} \in [0,1]^{d \times c} : \mathbf{M}^\mathrm{T}\mathbb{1}_d = \mathbb{1}_c\}$. The number of classes is $c$, and the number of possible partial label vectors is $d \leq 2^c$.

### 2.2 Learning from partial labels

Let $\mathcal{X}$ be a sample set, $\mathcal{Y} = \{\mathbf{e}_j^c, j = 0, 1, \ldots, c-1\}$, a set of labels, and $\mathcal{Z} \subset \{0,1\}^c$ a set of partial labels. Sample $(\mathbf{x}, \mathbf{z}) \in \mathcal{X} \times \mathcal{Z}$ is drawn from an unknown distribution $P$.

Partial label vector $\mathbf{z} \in \mathcal{Z}$ is a noisy version of the true label $\mathbf{y} \in \mathcal{Y}$. Several authors [1] [6] [7] [8] assume that the true label is always present in $\mathbf{z}$, i.e., $z_j = 1$ when $y_j = 1$, but this assumption is not required in our setting, which admits noisy label scenarios (as, for instance, in [2]). Without loss of generality, we assume that $\mathcal{Z}$ contains only partial labels with nonzero probability (i.e. $P\{\mathbf{z} = \mathbf{b}\} > 0$ for any $\mathbf{b} \in \mathcal{Z}$).

In general, we model the relationship between $\mathbf{z}$ and $\mathbf{y}$ through an arbitrary $d \times c$ conditional mixing probability matrix $\mathbf{M}(\mathbf{x})$ with components

$$m_{ij}(\mathbf{x}) = P\{\mathbf{z} = \mathbf{b}_i | y_j = 1, \mathbf{x}\} \tag{1}$$

where $\mathbf{b}_i \in \mathcal{Z}$ is the $i$-th element of $\mathcal{Z}$ for some arbitrary ordering.

Note that, in general, the mixing matrix could depend on $\mathbf{x}$, though a constant mixing matrix [2] [6] [7] [8] is a common assumption, as well as the statistical independence of the incorrect labels [6] [7] [8]. In this paper we do not impose these assumptions.

The goal is to infer $\mathbf{y}$ given $\mathbf{x}$ without knowing model $P$. To do so, a set of partially labelled samples, $\mathcal{S} = \{(\mathbf{x}_k, \mathbf{z}_k), k = 1, \ldots, K\}$ is available. True labels $\mathbf{y}_k$ are not observed.

We will illustrate different partial label scenarios with a 3-class problem. Consider that each column of $\mathbf{M}^\mathrm{T}$ corresponds to a label pattern $(z_0, z_1, z_2)$ following the ordering $(0,0,0)$, $(1,0,0)$, $(0,1,0)$, $(0,0,1)$, $(1,1,0)$, $(1,0,1)$, $(0,1,1)$, $(1,1,1)$ (e.g. the first column contains $P\{\mathbf{z} = (0,0,0)^\mathrm{T}|y_j = 1\}$, for $j = 0, 1, 2$).

A. Supervised learning: $\mathbf{M} = \begin{pmatrix} 0 & 1 & 0 & 0 & 0 & 0 & 0 & 0 \\ 0 & 0 & 1 & 0 & 0 & 0 & 0 & 0 \\ 0 & 0 & 0 & 1 & 0 & 0 & 0 & 0 \end{pmatrix}^\mathrm{T}$

B. Single noisy labels: $\mathbf{M} = \begin{pmatrix} 0 & 1-\alpha & \alpha/2 & \alpha/2 & 0 & 0 & 0 & 0 \\ 0 & \beta/2 & 1-\beta & \beta/2 & 0 & 0 & 0 & 0 \\ 0 & \gamma/2 & \gamma/2 & 1-\gamma & 0 & 0 & 0 & 0 \end{pmatrix}^\mathrm{T}$

C. Semisupervised learning: $\mathbf{M} = \begin{pmatrix} \alpha & 1-\alpha & 0 & 0 & 0 & 0 & 0 & 0 \\ \beta & 0 & 1-\beta & 0 & 0 & 0 & 0 & 0 \\ \gamma & 0 & 0 & 1-\gamma & 0 & 0 & 0 & 0 \end{pmatrix}^\mathrm{T}$

D. True label with independent noisy labels:

$\mathbf{M} = \begin{pmatrix} 0 & 1-\alpha-\alpha^2 & 0 & 0 & \alpha/2 & \alpha/2 & 0 & \alpha^2 \\ 0 & 0 & 1-\beta-\beta^2 & 0 & \beta/2 & 0 & \beta/2 & \beta^2 \\ 0 & 0 & 0 & 1-\gamma-\gamma^2 & 0 & \gamma/2 & \gamma/2 & \gamma^2 \end{pmatrix}^\mathrm{T}$

E. Two labels, one of them true: $\mathbf{M} = \begin{pmatrix} 0 & 1-\alpha & 0 & 0 & \alpha/2 & \alpha/2 & 0 & 0 \\ 0 & 0 & 1-\beta & 0 & \beta/2 & 0 & \beta/2 & 0 \\ 0 & 0 & 0 & 1-\gamma & 0 & \gamma/2 & \gamma/2 & 0 \end{pmatrix}^{\mathrm{T}}$

The question that motivates our work is the following: knowing $\mathbf{M}$ (i.e. knowing the scenario and the value of parameters $\alpha$, $\beta$ and $\gamma$), we can estimate accurate posterior class probabilities from partially labelled data in all these cases, however, is it possible if $\alpha$, $\beta$ and $\gamma$ are unknown? We will see that the answer is negative for scenarios B,C,D, but it is positive for E. In the positive case, no information is lost by the partial label process for infinite sample sizes. In the negative case, some performance is lost as a consequence of the mixing process, that persists even for infinite sample sizes [1]

## 2.3 Inference through partial label probabilities

If the mixing matrix is known, a conceptually simple strategy to solve the partial label problem consists of estimating posterior partial label probabilities, using them to estimate posterior class probabilities and predict $\mathbf{y}$. Since

$$P\{\mathbf{z} = \mathbf{b}_i|\mathbf{x}\} = \sum_{j=0}^{c-1} m_{ij}(\mathbf{x})P\{y_j = 1|\mathbf{x}\}, \qquad (2)$$

we can define vectors $\mathbf{p}(\mathbf{x})$ and $\boldsymbol{\eta}(\mathbf{x})$ with components $p_i = P\{\mathbf{z} = \mathbf{b}_i|\mathbf{x}\}$ and $\eta_j = P\{y_j = 1|\mathbf{x}\}$, to write (2) as $\mathbf{p}(\mathbf{x}) = \mathbf{M}(\mathbf{x})\boldsymbol{\eta}(\mathbf{x})$ and, thus,

$$\boldsymbol{\eta}(\mathbf{x}) = \mathbf{M}^+(\mathbf{x})\mathbf{p}(\mathbf{x}) \qquad (3)$$

where $\mathbf{M}^+(\mathbf{x}) = (\mathbf{M}^{\mathrm{T}}(\mathbf{x})\mathbf{M}(\mathbf{x}))^{-1}\mathbf{M}^{\mathrm{T}}(\mathbf{x})$ is the left inverse (pseudoinverse) of $\mathbf{M}(\mathbf{x})$.

Thus, a first condition to estimate $\boldsymbol{\eta}$ from $\mathbf{p}$ given $\mathbf{M}$ is that the conditional mixing matrix has a left inverse (i.e., the columns of $\mathbf{M}(\mathbf{x})$ are linearly independent).

There are some trivial cases where the mixing matrix has no pseudoinverse (for instance, if $P\{\mathbf{z}|\mathbf{y},\mathbf{x}\} = P\{\mathbf{z}|\mathbf{x}\}$, all rows in $\mathbf{M}(\mathbf{x})$ are equal, and $\mathbf{M}^{\mathrm{T}}(\mathbf{x})\mathbf{M}(\mathbf{x})$ is a rank 1 matrix, which has no inverse), but these are degenerate cases of no practical interest. From a practical point of view, the application of (3) states two major problems: (1) when the model $P$ is unknown, even knowing $\mathbf{M}$, estimating $\mathbf{p}$ from data may be infeasible for $d$ close to $2^c$ and a large number of classes (furthermore, posterior probability estimates will not be accurate if the sample size is small), and (2) $\mathbf{M}(\mathbf{x})$ is generally unknown, and cannot be estimated from the partially labelled set, $\mathcal{S}$.

The solution adopted in this paper for the first problem consists of estimating $\boldsymbol{\eta}$ from data without estimating $\mathbf{p}$. This is discussed in the next section. The second problem is discussed in Section 4.

# 3 Loss functions for posterior probability estimation

The estimation of posterior probabilities from labelled data is a well known problem in statistics and machine learning, that has received some recent attention in the machine learning literature [9] [10]. In order to estimate posteriors from labelled data, a loss function $\tilde{\ell}(\mathbf{y}, \hat{\boldsymbol{\eta}})$ is required such that $\boldsymbol{\eta}$ is a member of $\arg\min_{\hat{\boldsymbol{\eta}}} \mathbb{E}_{\mathbf{y}}\{\tilde{\ell}(\mathbf{y}, \hat{\boldsymbol{\eta}})\}$. Losses satisfying this property are said to be Fisher consistent and are known as proper scoring rules. A loss is strictly proper if $\boldsymbol{\eta}$ is the only member of this set. A loss is *regular* if it is finite for any $\mathbf{y}$, except possibly that $\tilde{\ell}(\mathbf{y}, \hat{\boldsymbol{\eta}}) = \infty$ if $y_j = 1$ and $\hat{\eta}_j = 0$.

Proper scoring rules can be characterized by the Savage's representation [11] [12]

**Theorem 3.1** *A regular scoring rule* $\tilde{\ell} : \mathcal{Z} \times \mathcal{P}_c \to \mathbb{R}$ *is (strictly) proper if and only if*

$$\tilde{\ell}(\mathbf{y}, \hat{\boldsymbol{\eta}}) = h(\hat{\boldsymbol{\eta}}) + \mathbf{g}(\hat{\boldsymbol{\eta}})(\mathbf{y} - \hat{\boldsymbol{\eta}}) \qquad (4)$$

*where $h$ is a (strictly) concave function and $\mathbf{g}(\hat{\boldsymbol{\eta}})$ is a supergradient of $h$ at the point $\hat{\boldsymbol{\eta}}$, for all $\hat{\boldsymbol{\eta}} \in \mathcal{P}_c$.*

(Remind that $\mathbf{g}$ is a supergradient of $h$ at $\hat{\boldsymbol{\eta}}$ if $h(\boldsymbol{\eta}) \leq h(\hat{\boldsymbol{\eta}}) + \mathbf{g}^\mathsf{T}(\boldsymbol{\eta} - \hat{\boldsymbol{\eta}})$).

In order to deal with partial labels, we generalize proper losses as follows

**Definition** Let $\mathbf{y}$ and $\mathbf{z}$ be random vectors taking values in $\mathcal{Y}$ and $\mathcal{Z}$, respectively. A scoring rule $\ell(\mathbf{z}, \hat{\boldsymbol{\eta}})$ is proper to estimate $\boldsymbol{\eta}$ (with components $\eta_j = P\{y_j = 1\}$) from $\mathbf{z}$ if

$$\boldsymbol{\eta} \in \arg\min_{\hat{\boldsymbol{\eta}}} \mathbb{E}_{\mathbf{z}}\{\ell(\mathbf{z}, \hat{\boldsymbol{\eta}})\} \tag{5}$$

It is strictly proper if $\boldsymbol{\eta}$ is the only member of this set.

This generalized family of proper scoring rules can be characterized by the following.

**Theorem 3.2** *Scoring rule $\ell(\mathbf{z}, \hat{\boldsymbol{\eta}})$ is (strictly) proper to estimate $\boldsymbol{\eta}$ from $\mathbf{z}$ if and only if the* equivalent *loss*

$$\tilde{\ell}(\mathbf{y}, \hat{\boldsymbol{\eta}}) = \mathbf{y}^T \mathbf{M}^T \mathbf{l}(\hat{\boldsymbol{\eta}}), \tag{6}$$

*where $\mathbf{l}(\hat{\boldsymbol{\eta}})$ is a vector with components $\ell_i(\hat{\boldsymbol{\eta}}) = \ell(\mathbf{b}_i, \hat{\boldsymbol{\eta}})$ and $\mathbf{b}_i$ is the $i$-th element in $\mathcal{Z}$ (according to some arbitrary ordering), is (strictly) proper.*

**Proof** The proof is straightforward by noting that the expected loss can be expressed as

$$\mathbb{E}_{\mathbf{z}}\{\ell(\mathbf{z}, \hat{\boldsymbol{\eta}})\} = \sum_{i=0}^{d-1} P\{\mathbf{z} = \mathbf{b}_i\} \ell_i(\hat{\boldsymbol{\eta}}) = \sum_{i=0}^{d-1} \sum_{j=0}^{c-1} m_{ij} \eta_j \ell_i(\hat{\boldsymbol{\eta}})$$

$$= \boldsymbol{\eta}^\mathsf{T} \mathbf{M}^\mathsf{T} \mathbf{l}(\hat{\boldsymbol{\eta}}) = \mathbb{E}_{\mathbf{y}}\{\mathbf{y}^\mathsf{T} \mathbf{M}^\mathsf{T} \mathbf{l}(\hat{\boldsymbol{\eta}})\} = \mathbb{E}_{\mathbf{y}}\{\tilde{\ell}(\mathbf{y}, \hat{\boldsymbol{\eta}})\} \tag{7}$$

Therefore, $\arg\min_{\hat{\boldsymbol{\eta}}} \mathbb{E}_{\mathbf{z}}\{\ell(\mathbf{z}, \hat{\boldsymbol{\eta}})\} = \arg\min_{\hat{\boldsymbol{\eta}}} \mathbb{E}_{\mathbf{y}}\{\tilde{\ell}(\mathbf{y}, \hat{\boldsymbol{\eta}})\}$ and, thus, $\ell$ is (strictly) proper with respect to $\mathbf{y}$ iff $\tilde{\ell}$ is (strictly) proper.

Note that, defining vector $\tilde{\mathbf{l}}(\hat{\boldsymbol{\eta}})$ with components $\tilde{\ell}_j(\hat{\boldsymbol{\eta}}) = \tilde{\ell}(\mathbf{e}_j^c, \hat{\boldsymbol{\eta}})$, we can write

$$\tilde{\mathbf{l}}(\hat{\boldsymbol{\eta}}) = \mathbf{M}^\mathsf{T} \mathbf{l}(\hat{\boldsymbol{\eta}}) \tag{8}$$

We will use this vector representation of losses extensively in the following.

Th. 3.2 states that the proper character of a loss for estimating $\boldsymbol{\eta}$ from $\mathbf{z}$ depends on $\mathbf{M}$. For this reason, in the following we will say that $\ell(\mathbf{z}, \hat{\boldsymbol{\eta}})$ is $\mathbf{M}$-proper if it is proper to estimate $\boldsymbol{\eta}$ from $\mathbf{z}$.

## 4   Proper losses for sets of mixing matrices

Eq. (8) may be useful to check if a given loss is $\mathbf{M}$-proper. However, note that, since matrix $\mathbf{M}^T$ is $d \times c$, it has no left inverse, and we cannot take $\mathbf{M}^\mathsf{T}$ out from the left side of (8) to compute $\ell$ from $\tilde{\ell}$. For any given $\mathbf{M}$ and any given equivalent loss $\tilde{\mathbf{l}}(\hat{\boldsymbol{\eta}})$, there is an uncountable number of losses $\mathbf{l}(\hat{\boldsymbol{\eta}})$ satisfying (8).

**Example** Let $\tilde{\ell}$ be an arbitrary proper loss for a 3-class problem. The losses

$$\ell(\mathbf{z}, \hat{\boldsymbol{\eta}}) = (z_0 - z_1 z_2)\tilde{\ell}_0(\hat{\boldsymbol{\eta}}) + (z_1 - z_0 z_2)\tilde{\ell}_1(\hat{\boldsymbol{\eta}}) + (z_2 - z_0 z_1)\tilde{\ell}_2(\hat{\boldsymbol{\eta}}) \tag{9}$$

$$\ell'(\mathbf{z}, \hat{\boldsymbol{\eta}}) = z_0 \tilde{\ell}_0(\hat{\boldsymbol{\eta}}) + z_1 \tilde{\ell}_1(\hat{\boldsymbol{\eta}}) + z_2 \tilde{\ell}_2(\hat{\boldsymbol{\eta}}) \tag{10}$$

are $\mathbf{M}$-proper for the mixing matrix $\mathbf{M}$ given by

$$m_{ij} = \left[ \begin{array}{ll} 1 & \text{if } \mathbf{b}_i = \mathbf{e}_j^c \\ 0 & \text{otherwise} \end{array} \right. \tag{11}$$

Note that $\mathbf{M}$ corresponds to a situation where labels are perfectly labelled, and $\mathbf{z}$ contains perfect information about $\mathbf{y}$ (in fact, $\mathbf{z} = \mathbf{y}$ with probability one).

Also, for any $\ell(\mathbf{z}, \hat{\boldsymbol{\eta}})$, there are different mixing matrices such that the equivalent loss is the same.

**Example** The loss given by (9) is $\mathbf{M}$-proper for the mixing matrix $\mathbf{M}$ in (11) and it is also $\mathbf{N}$-proper, for $\mathbf{N}$ with components

$$n_{ij} = \left[ \begin{array}{ll} 1/2 & \text{if } \mathbf{b}_i = \mathbf{e}_j^c + \mathbf{e}_k^c, \text{ for some } k \neq j \\ 0 & \text{otherwise} \end{array} \right. \tag{12}$$

Matrix $\mathbf{N}$ corresponds to a situation where label $\mathbf{z}$ contains the true class and another noisy component taken at random from the other classes.

In general, if $\mathbf{l}(\hat{\boldsymbol{\eta}})$ is $\mathbf{M}$-proper and $\mathbf{N}$-proper with equivalent loss $\tilde{\mathbf{l}}(\hat{\boldsymbol{\eta}})$, then it is also $\mathbf{Q}$-proper with the same equivalent loss, for any $\mathbf{Q}$ in the form

$$\mathbf{Q} = \mathbf{M}(\mathbf{I} - \mathbf{D}) + \mathbf{N}\mathbf{D} \tag{13}$$

where $\mathbf{D}$ is a diagonal nonnegative matrix (note that $\mathbf{Q}$ is a probability matrix, because $\mathbf{Q}^{\mathsf{T}}\mathbb{1}_d = \mathbb{1}_c$). This is because

$$\mathbf{Q}^{\mathsf{T}}\mathbf{l}(\hat{\boldsymbol{\eta}}) = (\mathbf{I} - \mathbf{D})\mathbf{M}^{\mathsf{T}}\mathbf{l}(\hat{\boldsymbol{\eta}}) + \mathbf{D}\mathbf{N}^{\mathsf{T}}\mathbf{l}(\hat{\boldsymbol{\eta}}) = (\mathbf{I} - \mathbf{D})\tilde{\mathbf{l}}(\hat{\boldsymbol{\eta}}) + \mathbf{D}\tilde{\mathbf{l}}(\hat{\boldsymbol{\eta}}) = \tilde{\mathbf{l}}(\hat{\boldsymbol{\eta}}) \tag{14}$$

More generally, for arbitrary non-diagonal matrices $\mathbf{D}$, provided that $\mathbf{Q}$ is a probability matrix, $\mathbf{l}(\hat{\boldsymbol{\eta}})$ is $\mathbf{Q}$-proper.

**Example** Assuming diagonal $\mathbf{D}$, if $\mathbf{M}$ and $\mathbf{N}$ are the mixing matrices defined in (11) and (12), respectively, the loss (9) is $\mathbf{Q}$-proper for any mixing matrix $\mathbf{Q}$ in the form (13). This corresponds to a matrix with components

$$q_{ij} = \left[ \begin{array}{ll} d_{jj} & \text{if } \mathbf{b}_i = \mathbf{e}_j^c \\ (1 - d_{jj})/2 & \text{if } \mathbf{b}_i = \mathbf{e}_j^c + \mathbf{e}_k^c, \text{ for some } k \neq j \\ 0 & \text{otherwise} \end{array} \right. \tag{15}$$

That is, the loss in (9) is proper for any situation where the label $\mathbf{z}$ contains the true class and possibly another class taken at random, and the probability that the true label is corrupted may be class-dependent.

## 4.1 Building proper losses from ambiguity sets

The ambiguity on $\mathbf{M}$ for a given loss $\mathbf{l}(\hat{\boldsymbol{\eta}})$ can be used to deal with the second problem mentioned in Sec. 2.3: in general, the mixing matrix may be unknown, or, even if it is known, it may depend on the observation, $\mathbf{x}$. Thus, we need a procedure to design losses that are proper for a wide family of mixing matrices. In general, given a set of mixing matrices, $\mathcal{Q}$, we will say that $\ell$ is $\mathcal{Q}$-proper if it is $\mathbf{M}$-proper for any $\mathbf{M} \in \mathcal{Q}$

The following result provides a way to construct a proper loss $\ell$ for partial labels from a given conventional proper loss $\tilde{\ell}$.

**Theorem 4.1** *For $0 \leq j \leq c - 1$, let $\mathcal{V}_j = \{\mathbf{v}_i^j \in \mathcal{P}_d, 1 \leq i \leq n_j\}$ be a set of $n_j > 0$ probability vectors with dimension $d$, such that $\sum_{j=0}^{c-1} n_j = d$ and $span(\cup_{j=0}^{c-1}\mathcal{V}_j) = \mathbb{R}^d$ and let $\mathcal{Q} = \{\mathbf{M} \in \mathcal{M} : \mathbf{M}\mathbf{e}_j^c \in span(\mathcal{V}_j) \cap \mathcal{P}_d\}$.*

*Then, for any (strictly) proper loss $\tilde{\ell}(\mathbf{y}, \hat{\boldsymbol{\eta}})$, there exists a loss $\ell(\mathbf{z}, \hat{\boldsymbol{\eta}})$ which is (strictly) $\mathcal{Q}$-proper.*

**Proof** The proof is constructive. Let $\mathbf{V}$ be a $d \times d$ matrix whose columns are the elements of $\cup_{j=0}^{c-1}\mathcal{V}_j$, which is invertible since $span(\cup_{j=0}^{c-1}\mathcal{V}_j) = \mathbb{R}^d$. Let $\mathbf{c}(\hat{\boldsymbol{\eta}})$ be a $d \times 1$ vector such that $c_i(\hat{\boldsymbol{\eta}}) = \tilde{\ell}_j(\hat{\boldsymbol{\eta}})$ if $\mathbf{V}\mathbf{e}_i^d \in \mathcal{V}_j$.

Let $\ell(\mathbf{z}, \hat{\boldsymbol{\eta}})$ be a loss defined by vector $\mathbf{l}(\hat{\boldsymbol{\eta}}) = (\mathbf{V}^{\mathsf{T}})^{-1}\mathbf{c}(\hat{\boldsymbol{\eta}})$.

Consider the set $\mathcal{R} = \{\mathbf{M} \in \mathcal{M} : \mathbf{M}\mathbf{e}_j^c \in \mathcal{V}_j \text{ for all } j\}$ (which is not empty because $n_j > 0$). Since the columns of any $\mathbf{M} \in \mathcal{R}$ are also columns of $\mathbf{V}$, then $\mathbf{M}^{\mathsf{T}}\mathbf{l}(\hat{\boldsymbol{\eta}}) = \tilde{\mathbf{l}}(\hat{\boldsymbol{\eta}})$ and, thus, $\ell(\mathbf{z}, \hat{\boldsymbol{\eta}})$ is $\mathbf{M}$-proper. Therefore, it is also proper for any affine combination of matrices in $\mathcal{R}$ inside $\mathcal{P}_d$. But $span(\mathcal{R}) \cap \mathcal{P}_d = \mathcal{Q}$. Thus, $\ell(\mathbf{z}, \hat{\boldsymbol{\eta}})$ is $\mathbf{M}$-proper for all $\mathbf{M} \in \mathcal{Q}$ (i.e. it is $\mathcal{Q}$-proper).

Theorem 4.1 shows that we can construct proper losses for learning from partial labels by specifying the points of sets $\mathcal{V}_j$, $j = 0, \ldots, c-1$. Each of these sets defines an *ambiguity set* $\mathcal{A}_j = \text{span}(\mathcal{V}_j) \cap \mathcal{P}_d$ which represents all admissible conditional distributions for $P(\mathbf{z}|y_j = 1)$. If the columns of the true mixing matrix $\mathbf{M}$ are members of the ambiguity set, the resulting loss can be used to estimate posterior class probabilities from the observed partial labels.

Thus, a general procedure to design a loss function for learning from partial labels is:

1. Select a proper loss, $\tilde{\ell}(\mathbf{y}, \hat{\boldsymbol{\eta}})$
2. Define the ambiguity sets by choosing, for each class $j$, a set $\mathcal{V}_j$ of $n_j$ linearly independent basis vectors for each class. The whole set of $d$ basis vectors must be linearly independent.
3. Construct matrix $\mathbf{V}$ whose columns comprise all basis vectors.
4. Construct binary matrix $\mathbf{U}$ with $u_{ji} = 1$ if the $i$-th column of $\mathbf{V}$ is in $\mathcal{V}_j$, and $u_{ji} = 0$ otherwise.
5. Compute the desired proper loss vector as

$$\mathbf{l}(\hat{\boldsymbol{\eta}}) = (\mathbf{V}^{\mathrm{T}})^{-1} \mathbf{U} \tilde{\mathbf{l}}(\hat{\boldsymbol{\eta}}) \tag{16}$$

Since the ambiguity set $\mathcal{A}_j$ is the intersection of a $n_j$-dimensional linear subspace with the $d$-dimensional probability simplex, it is a $n_j - 1$ dimensional convex polytope whose vertices lie in distinct $(n_j - 1)$-faces of $\mathcal{P}_d$. These vertices must have a set of at least $n_j - 1$ zero components which cannot be a set of zeros in any other vertex.

This has two consequences: (1) we can define the ambiguity sets from these vertices, and (2), the choice is not unique, because the number of vertices can be higher than $n_j - 1$.

If proper loss $\tilde{\ell}(\mathbf{y}, \hat{\boldsymbol{\eta}})$ is non degenerate, $\mathcal{Q}$ contains all mixing matrices for which a loss is proper:

**Theorem 4.2** *Let us assume that, if $\mathbf{a}^T \tilde{\mathbf{l}}(\hat{\boldsymbol{\eta}}) = 0$ for any $\hat{\boldsymbol{\eta}}$, then $\mathbf{a} = \mathbf{0}$. Under the conditions of Theorem 4.1, for any $\mathbf{M} \in \mathcal{M} \setminus \mathcal{Q}$, $\ell(\mathbf{z}, \hat{\boldsymbol{\eta}})$ is not $\mathbf{M}$-proper.*

**Proof** Since the columns of $\mathbf{V}$ are in the ambiguity sets and form a basis of $\mathbb{R}^d$, $\text{span}(\cup_{j=0}^{c-1} \mathcal{A}_j) = \mathbb{R}^d$. Thus, the $n$-th column of any arbitrary $\mathbf{M}$ can be represented as $\mathbf{m}_n = \sum_{j=1}^{c} \alpha_{n,j} \mathbf{w}_j$ for some $\mathbf{w}_j \in \mathcal{A}_j$ and some coefficients $\alpha_{nj}$. If $\mathbf{M} \notin \mathcal{Q}$, $\alpha_{nj} \neq 0$ for some $j \neq n$ and some $n$. Then $\mathbf{m}_n^{\mathsf{T}} \mathbf{l}(\hat{\boldsymbol{\eta}}) = \sum_{j=1}^{l} \alpha_{nj} \tilde{\ell}_j(\hat{\boldsymbol{\eta}})$, which cannot be equal to $\ell_n(\hat{\boldsymbol{\eta}})$ for all $\hat{\boldsymbol{\eta}}$. Therefore, $\ell(\mathbf{z}, \hat{\boldsymbol{\eta}})$ is not $\mathbf{M}$-proper.

### 4.2 Virtual labels

The analysis above shows a procedure to construct proper losses from ambiguity sets. The main result of this section is to show that (16) is actually a universal representation, in the sense that any proper loss can be represented in this form, and we generalize the Savage's representation providing and explicit formula for $\mathcal{Q}$-proper losses.

**Theorem 4.3** *Scoring rule $\ell(\mathbf{z}, \hat{\boldsymbol{\eta}})$ is (strictly) $\mathcal{Q}$-proper for some matrix set $\mathcal{Q}$ with equivalent loss $\tilde{\ell}(\mathbf{y}, \hat{\boldsymbol{\eta}})$ if and only if*

$$\ell(\mathbf{z}, \hat{\boldsymbol{\eta}}) = h(\hat{\boldsymbol{\eta}}) + \mathbf{g}(\hat{\boldsymbol{\eta}})^T (\mathbf{U}^T \mathbf{V}^{-1} \mathbf{z} - \hat{\boldsymbol{\eta}}). \tag{17}$$

*where $h$ is the (strictly) concave function from the Savage's representation for $\tilde{\ell}$, $\mathbf{g}(\hat{\boldsymbol{\eta}})$ is a supergradient of $h$, $\mathbf{V}$ is a $d \times d$ non-singular matrix and $\mathbf{U}$ is a binary matrix with only one unit value at each row.*

*Moreover, the ambiguity set of class $j$ is $\mathcal{A}_j = \text{span}(\mathcal{V}_j)$, where $\mathcal{V}_j$ is the set of all columns in $\mathbf{V}$ such that $u_{ji} = 1$.*

**Proof** See the Appendix.

Comparing (4) with (17), the effect of imperfect labelling becomes clear: the unknown true label $\mathbf{y}$ is replaced by a *virtual* label $\tilde{\mathbf{y}} = \mathbf{U}^{\mathsf{T}} \mathbf{V}^{-1} \mathbf{z}$, which is a linear combination of the partial labels.

### 4.3 Admissible scenarios

The previous analysis shows that, in order to calibrate posterior probabilities from partial labels in scenarios where the mixing matrix is known, two conditions are required: (1) the rows of any admissible mixing matrix must be contained in the admissible sets, (2) the basis of all admissible sets must be linearly independent. It is not difficult to see that the parametric matrices in scenarios B, C and D defined in Section 2.2 cannot be generated using a set of basis satisfying these constraints. On the contrary, scenario E is admissible, as we have shown in the example in Section 4.

## 5 Estimation Errors

If the true mixing matrix $\mathbf{M}$ is not in $\mathcal{Q}$, a $\mathcal{Q}$-proper loss may fail to estimate $\boldsymbol{\eta}$. The consequences of this can be analyzed using the expected loss, given by

$$L(\boldsymbol{\eta}, \hat{\boldsymbol{\eta}}) \doteq \mathbb{E}\{\ell(\mathbf{z}, \hat{\boldsymbol{\eta}})\} = \boldsymbol{\eta}^{\mathrm{T}} \mathbf{M}^{\mathrm{T}} \mathbf{l}(\hat{\boldsymbol{\eta}}) = \boldsymbol{\eta}^{\mathrm{T}} \mathbf{M}^{\mathrm{T}} (\mathbf{V}^{\mathrm{T}})^{-1} \mathbf{U} \tilde{\mathbf{l}}(\hat{\boldsymbol{\eta}}) \tag{18}$$

If $\mathbf{M} \in \mathcal{Q}$, then $L(\boldsymbol{\eta}, \hat{\boldsymbol{\eta}}) = \boldsymbol{\eta}^{\mathrm{T}} \tilde{\mathbf{l}}(\hat{\boldsymbol{\eta}})$. However, if $\mathbf{M} \notin \mathcal{Q}$, then we can decompose $\mathbf{M} = \mathbf{M}_{\mathcal{Q}} + \mathbf{N}$, where $\mathbf{M}_{\mathcal{Q}}$ is the orthogonal projection of $\mathbf{M}$ in $\mathcal{Q}$. Then

$$L(\boldsymbol{\eta}, \hat{\boldsymbol{\eta}}) = \boldsymbol{\eta}^{\mathrm{T}} \mathbf{N}^{\mathrm{T}} (\mathbf{V}^{\mathrm{T}})^{-1} \mathbf{U} \tilde{\mathbf{l}}(\hat{\boldsymbol{\eta}}) + \boldsymbol{\eta}^{\mathrm{T}} \tilde{\mathbf{l}}(\hat{\boldsymbol{\eta}}) \tag{19}$$

**Example** The effect of a bad choice of the ambiguity set can be illustrated using the loss in (9) in two cases: $\tilde{l}_j(\hat{\boldsymbol{\eta}}) = \|\mathbf{e}_j^c - \hat{\boldsymbol{\eta}}\|^2$ (the square error) and $\tilde{l}_j(\hat{\boldsymbol{\eta}}) = -\ln(\hat{\eta}_j)$ (the cross entropy). As we have discussed before, loss (9) is proper for any scenario where label $\mathbf{z}$ contains the true class and possibly another class taken at random. Let us assume that the true mixing matrix is

$$\mathbf{M} = \begin{pmatrix} 0.5 & 0 & 0 & 0.4 & 0.1 & 0 \\ 0 & 0.5 & 0 & 0.3 & 0 & 0.2 \\ 0 & 0 & 0.6 & 0 & 0.2 & 0.2 \end{pmatrix}^{\mathrm{T}} \tag{20}$$

(were each column of $\mathbf{M}^{\mathrm{T}}$ corresponds to a label vector $(z_0, z_1, z_2)$ following the ordering $(1,0,0)$, $(0,1,0)$, $(0,0,1)$, $(1,1,0)$, $(1,0,1)$, $(0,1,1)$. Fig. 1 shows the expected loss in (19) for the square error (left) and the cross entropy (right), as a function of $\hat{\boldsymbol{\eta}}$ over the probability simplex $\mathcal{P}_3$, for $\boldsymbol{\eta} = (0.45, 0.15, 0.4)^{\mathrm{T}}$. Since $\mathbf{M} \notin \mathcal{Q}$, the estimated posterior minimizing expected loss, $\hat{\eta}^*$ (which is unique because both losses are strictly proper), does not coincide with the true posterior.

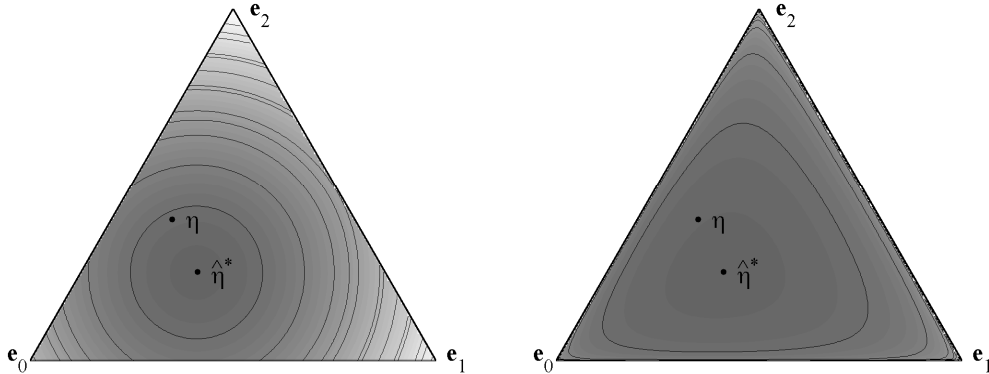

Figure 1: Square loss (left) and cross entropy (right) in the probability simplex, as a function of $\hat{\boldsymbol{\eta}}$ for $\boldsymbol{\eta} = (0.45, 0.15, 0.4)^{\mathrm{T}}$

It is important to note that the minimum $\hat{\eta}^*$ does not depend on the choice of the cost and, thus, the estimation error is invariant to the choice of the strict proper loss (though this could be not true when $\boldsymbol{\eta}$ is estimated from an empirical distribution). This is because, using (19) and noting that the expected proper loss is

$$\tilde{L}(\boldsymbol{\eta}, \hat{\boldsymbol{\eta}}) \doteq \mathbb{E}_{\mathbf{y}} \tilde{\ell}(\mathbf{y}, \hat{\boldsymbol{\eta}}) = \boldsymbol{\eta}^{\mathrm{T}} \tilde{\mathbf{l}}(\hat{\boldsymbol{\eta}}) \tag{21}$$

we have
$$L(\boldsymbol{\eta}, \hat{\boldsymbol{\eta}}) = L(\mathbf{U}^{\mathrm{T}}\mathbf{V}^{-1}\mathbf{M}\boldsymbol{\eta}, \hat{\boldsymbol{\eta}}) \tag{22}$$
Since (22) is minimum for $\hat{\boldsymbol{\eta}}^* = \mathbf{U}^{\mathrm{T}}\mathbf{V}^{-1}\mathbf{M}\boldsymbol{\eta}$, the estimation error is
$$\|\boldsymbol{\eta} - \hat{\boldsymbol{\eta}}^*\|^2 = \|(\mathbf{I} - \mathbf{U}^{\mathrm{T}}\mathbf{V}^{-1}\mathbf{M})\boldsymbol{\eta}\|^2 \tag{23}$$
which is independent on the particular choice of the equivalent loss.

If $\tilde{\ell}$ is proper but not strictly proper, the minimum may be not unique. For instance, for the $0-1$ loss, any $\hat{\boldsymbol{\eta}}$ providing the same decisions than $\boldsymbol{\eta}$ is a minimum of $\tilde{L}(\boldsymbol{\eta}, \hat{\boldsymbol{\eta}})$. Therefore, those values of $\boldsymbol{\eta}$ with $\boldsymbol{\eta}$ and $\mathbf{U}^{\mathrm{T}}\mathbf{V}^{-1}\mathbf{M}\boldsymbol{\eta}$ in the same decision region are not influenced by a bad choice of the ambiguity set. Unfortunately, since the set of boundary decision points is not linear (but piecewise linear) one can always find points $\boldsymbol{\eta}$ that are affected by this choice. Therefore, a wrong choice of the ambiguity set always changes the boundary decision. Summarizing, the ambiguity set for probability estimation is not larger than that for classification.

# 6   Conclusions

In this paper we have generalized proper losses to deal with scenarios with partial labels. Proper losses based on partial labels can be designed to cope with different mixing matrices. We have also generalized the Savage's representation of proper losses to obtain an explicit expression for proper losses as a function of a concave generator.

# Appendix: Proof of Theorem 4.3

Let us assume that $\ell(\mathbf{z}, \hat{\boldsymbol{\eta}})$ is (strictly) $\mathcal{Q}$-proper for some matrix set $\mathcal{Q}$ with equivalent loss $\tilde{\ell}(\mathbf{y}, \hat{\boldsymbol{\eta}})$. Let $\mathcal{Q}_j$ be the set of the $j$-th rows of all matrices in $\mathcal{Q}$, and take $\mathcal{A}_j = \mathrm{span}(\mathcal{Q}_j) \cap \mathcal{P}_d$. Then any vector $\mathbf{m} \in \mathcal{A}_j$ is affine combination of vectors in $\mathcal{Q}_j$ and, thus, $\mathbf{m}^{\mathrm{T}}\mathbf{l}(\hat{\boldsymbol{\eta}}) = \tilde{l}(\hat{\boldsymbol{\eta}})$. Therefore, if $\mathrm{span}(\mathcal{Q}_i)$ has dimension $n_i$, we can take a basis $\mathcal{V}_i \in \mathcal{Q}_i$ of $n_i$ linearly independent vectors such that $\mathcal{A}_i = \mathrm{span}(\mathcal{Q}_i) \cap \mathcal{P}_d$.

By construction $\mathbf{l}(\hat{\boldsymbol{\eta}}) = (\mathbf{V}^{\mathrm{T}})^{-1}\mathbf{U}\tilde{\mathbf{l}}(\hat{\boldsymbol{\eta}})$. Combining this equation with the Savage's representation in (4), we get
$$\begin{aligned} \ell(\mathbf{z}, \hat{\boldsymbol{\eta}}) = \mathbf{z}^{\mathrm{T}}\mathbf{l}(\hat{\boldsymbol{\eta}}) &= \mathbf{z}^{\mathrm{T}}(\mathbf{V}^{\mathrm{T}})^{-1}\mathbf{U}(h(\hat{\boldsymbol{\eta}})\mathbb{1}_c + (\mathbf{I} - \hat{\boldsymbol{\eta}}\mathbb{1}_c^{\mathrm{T}})^{\mathrm{T}}\mathbf{g}(\hat{\boldsymbol{\eta}}) \\ &= h(\hat{\boldsymbol{\eta}})\mathbf{z}^{\mathrm{T}}\mathbb{1}_d + \mathbf{z}^{\mathrm{T}}(\mathbf{V}^{\mathrm{T}})^{-1}(\mathbf{U} - \mathbb{1}_d\hat{\boldsymbol{\eta}}^{\mathrm{T}})\mathbf{g}(\hat{\boldsymbol{\eta}}) \\ &= h(\hat{\boldsymbol{\eta}}) + \mathbf{g}(\hat{\boldsymbol{\eta}})^{\mathrm{T}}(\mathbf{U}^{\mathrm{T}}\mathbf{V}^{-1}\mathbf{z} - \hat{\boldsymbol{\eta}}) \end{aligned} \tag{24}$$
which is the desired result.

Now, let us assume that (17) is true. Then
$$\mathbf{l}(\hat{\boldsymbol{\eta}}) = h(\hat{\boldsymbol{\eta}})\mathbb{1}_d + ((\mathbf{V}^{\mathrm{T}})^{-1}\mathbf{U} - \mathbb{1}_d\hat{\boldsymbol{\eta}}^{\mathrm{T}})\mathbf{g}(\hat{\boldsymbol{\eta}}). \tag{25}$$
For any matrix $\mathbf{M} \in \mathcal{M}$ such that $\mathbf{M}^{\mathrm{T}}\mathbf{e}_j^c \in \mathcal{A}_j$, we have
$$\mathbf{M}^{\mathrm{T}}\mathbf{l}(\hat{\boldsymbol{\eta}}) = h(\hat{\boldsymbol{\eta}})\mathbf{M}^{\mathrm{T}}\mathbb{1}_d + (\mathbf{M}^{\mathrm{T}}(\mathbf{V}^{\mathrm{T}})^{-1}\mathbf{U} - \mathbf{M}\mathbb{1}_d\hat{\boldsymbol{\eta}}^{\mathrm{T}})\mathbf{g}(\hat{\boldsymbol{\eta}}) \tag{26}$$
If $\mathbf{M} \in \mathcal{Q}$, then we can express each column, $j$, of $\mathbf{M}$ as a convex combination of the columns in $\mathbf{V}$ with $u_{ji} = 1$, thus $\mathbf{M} = \mathbf{V}\boldsymbol{\Lambda}$ for some matrix $\boldsymbol{\Lambda}$ with the coefficients of the convex combination at the corresponding positions of unit values in $\mathbf{U}$. Then $\mathbf{M}^{\mathrm{T}}(\mathbf{V}^{\mathrm{T}})^{-1}\mathbf{U} = \boldsymbol{\Lambda}\mathbf{U} = \mathbf{I}$. Using this in (26), we get
$$\mathbf{M}^{\mathrm{T}}\mathbf{l}(\hat{\boldsymbol{\eta}}) = h(\hat{\boldsymbol{\eta}})\mathbb{1}_c + (\mathbf{I}_c - \mathbb{1}_c\hat{\boldsymbol{\eta}}^{\mathrm{T}})\mathbf{g}(\hat{\boldsymbol{\eta}}) = \tilde{\mathbf{l}}(\hat{\boldsymbol{\eta}}). \tag{27}$$
Applying Theorem 3.2, the proof is complete.

# Acknowledgments

This work was partially funded by project TEC2011-22480 from the Spanish Ministry of Science and Innovation, project PRI-PIBIN-2011-1266 and by the IST Programme of the European Community, under the PASCAL2 Network of Excellence, IST-2007-216886. Thanks to Raúl Santos-Rodríguez and Darío García-García for their constructive comments about this manuscript

## Footnotes

[1] If the sample size is large (in particular for scenarios C and D), one could think of simply ignoring samples with imperfect labels, and training the classifier with the samples whose class is known. However, in general, there is some bias in this process, which eventually can degrade performance.

# References

[1] T. Cour, B. Sapp, and B. Taskar, "Learning from partial labels," *Journal of Machine Learning Research*, vol. 12, pp. 1225–1261, 2011.

[2] V. C. Raykar, S. Yu, L. H. Zhao, G. H. Valadez, C. Florin, L. Bogoni, and L. Moy, "Learning from crowds," *Journal of Machine Learning Research*, vol. 99, pp. 1297–1322, August 2010.

[3] V. S. Sheng, F. Provost, and P. G. Ipeirotis, "Get another label? improving data quality and data mining using multiple, noisy labelers," in *Procs. of the 14th ACM SIGKDD international conference on Knowledge discovery and data mining*, ser. KDD '08.   New York, NY, USA: ACM, 2008, pp. 614–622.

[4] E. Côme, L. Oukhellou, T. Denux, and P. Aknin, "Mixture model estimation with soft labels," in *Soft Methods for Handling Variability and Imprecision*, ser. Advances in Soft Computing, D. Dubois, M. Lubiano, H. Prade, M. Gil, P. Grzegorzewski, and O. Hryniewicz, Eds. Springer Berlin / Heidelberg, 2008, vol. 48, pp. 165–174.

[5] P. Liang, M. Jordan, and D. Klein, "Learning from measurements in exponential families," in *Proceedings of the 26th Annual International Conference on Machine Learning*.   ACM, 2009, pp. 641–648.

[6] R. Jin and Z. Ghahramani, "Learning with multiple labels," *Advances in Neural Information Processing Systems*, vol. 15, pp. 897–904, 2002.

[7] C. Ambroise, T. Denoeux, G. Govaert, and P. Smets, "Learning from an imprecise teacher: probabilistic and evidential approaches," in *Applied Stochastic Models and Data Analysis*, 2001, vol. 1, pp. 100–105.

[8] Y. Grandvalet and Y. Bengio, "Semi-supervised learning by entropy minimization," 2005.

[9] M. Reid and B. Williamson, "Information, divergence and risk for binary experiments," *Journal of Machine Learning Research*, vol. 12, pp. 731–817, 2011.

[10] H. Masnadi-Shirazi and N. Vasconcelos, "Risk minimization, probability elicitation, and cost-sensitive svms," in *Proceedings of the International Conference on Machine Learning*, 2010, pp. 204–213.

[11] L. Savage, "Elicitation of personal probabilities and expectations," *Journal of the American Statistical Association*, pp. 783–801, 1971.

[12] T. Gneiting and A. Raftery, "Strictly proper scoring rules, prediction, and estimation," *Journal of the American Statistical Association*, vol. 102, no. 477, pp. 359–378, 2007.

